# A Biologically Plausible Model for Rapid Natural Image Identification

**S. Ghebreab, A.W.M. Smeulders**
Intelligent Sensory Information Systems Group
University of Amsterdam, The Netherlands
s.ghebreab@uva.nl

**H. S. Scholte, V.A.F. Lamme**
Cognitive Neuroscience Group
University of Amsterdam, The Netherlands
h.s.scholte@uva.nl

## Abstract

Contrast statistics of the majority of natural images conform to a Weibull distribution. This property of natural images may facilitate efficient and very rapid extraction of a scene's visual gist. Here we investigated whether a neural response model based on the Weibull contrast distribution captures visual information that humans use to rapidly identify natural scenes. In a learning phase, we measured EEG activity of 32 subjects viewing brief flashes of 700 natural scenes. From these neural measurements and the contrast statistics of the natural image stimuli, we derived an across subject Weibull response model. We used this model to predict the EEG responses to 100 new natural scenes and estimated which scene the subject viewed by finding the best match between the model predictions and the observed EEG responses. In almost 90 percent of the cases our model accurately predicted the observed scene. Moreover, in most failed cases, the scene mistaken for the observed scene was visually similar to the observed scene itself. Similar results were obtained in a separate experiment in which 16 other subjects where presented with artificial occlusion models of natural images. Together, these results suggest that Weibull contrast statistics of natural images contain a considerable amount of visual gist information to warrant rapid image identification.

## 1 Introduction

Natural images, although apparently diverse, have a surprisingly regular statistical regularity. There is a strong correlation between adjacent image points in terms of local features such as luminance [1]. These second-order correlations decrease with distance between image points, giving rise to the typical $1/f^2$ power spectra of natural images. On account of this power-law characteristic, natural images compromise a very small and distinguishable subset of the space of all possible images, with specific scene categories occupying different parts of this subspace. For example, white noise images can be distinguished from natural images because of their deviation from the power law statistics, while street scenes and beach scenes can be separated from each other on the basis of differences in ensemble power spectra [2]. Thus, the power spectra of natural images contain an indeterminate amount of the visual gist of these images.

The similarity structure among nearby image points, however, represents only part of the statistical structure in natural images. There are also higher-order correlations, which introduce structure in the phase spectra of natural images. This structure is assumed to carry perceptually important image features such as edges and has been measured in terms of kurtosis in the contrast distribution of natural images [3, 4, 5]. Geusebroek and Smeulders [6] showed that the two-parameter Weibull distribution adequately captures the variance and kurtosis in the contrast distribution of the majority of natural images. In fact, the two parameters of the Weibull contrast distribution turn out to organize the space of all possible natural scenes in a perceptually meaningful manner [7] and thus are likely to provide additional information about a scene's visual gist.

Scholte et al. [7] have further shown that the two parameters of the Weibull contrast distribution match biologically realistic computations of Lateral Geniculate Nucleus (LGN) cells. Specifically, they simulated X-cell responses by filtering images with a difference of Gaussians (DoG), rectifying the filtered images and transforming the pixel values of the resulting images with a contrast gain function adequate for P-cells. To simulate Y-cell responses, the rectified images were passed through a Gaussian smoothing function and resulting pixel values were subsequently transformed with a contrast gain function adequate for M-cells. The sum of the resulting X-cell responses turned out to correlate highly with one Weibull parameter (r=0.95), whereas the sum of the resulting Y-cell responses correlated highly with the other Weibull parameter (r=0.70). Moreover, the two Weibull parameters correlated highly with EEG activity ($r^2$=0.5) at the occipital part of the brain. The findings of Scholte et al. [7] show that our brain is capable of approximating the Weibull contrast distribution of an image on the basis of filters that are biologically realistic in shape, sensitivity, and size.

Here we hypothesized that if Weibull contrast distributions of natural images carry perceptually important information, a neural response model based on the Weibull contrast distribution will predict brain responses to brief flashes of natural images. We tested this hypothesis with two experiments in which we rapidly presented a large set of natural or artificial images to multiple subjects while measuring EEG activity across the entire cortex. In each experiment, we constructed a neural response model from the Weibull statistics of the presented images and corresponding EEG data, which we then applied to predict EEG responses to a new collection of natural or artificial images. To validate the constructed neural response models, we used the approach of Kay et al. [8]: predicted and measured EEG responses were compared to determine whether the observed image was correctly identified.

## 2 Methods

We first describe how we filter images locally with a set of biologically-realistic filters. Then we address a local contrast response selection mechanism with which we construct a contrast magnitude map for a given input image (a detailed description is in submission [12]). Subsequently, Weibull contrast statistics are estimated from such maps and the relation between image statistics and neural activity modeled. The section ends with an explanation of a performance measure for EEG-based image identification.

### 2.1 Local contrasts values in natural images

As in [7], we use contrast filters that have spatial characteristics and contrast response properties closely mirroring well-known characteristic receptive-fields of LGN neurons [9]. Specifically, we use a bank of second-order Gaussian derivative filters that span multiple octaves in spatial scale, that have peak sensitivity approximately inverse to filter size and that have contrast gain properties independent of size. We represent contrast gain using an established non-linear response model that divides input contrast by the sum of the input and a semi-saturation constant [10]. In this model a low value of the semi-saturation parameter indicates high non-linear contrast gain whereas higher values result in a linear mapping and thus will not lead to saturation. Given an image, we process each image location with a bank of 5 contrast filters covering 5 octaves in spatial scale and, subsequently, subject the output of each scale-tuned filter to 5 different gain controls (5 semi-saturation values). This results, for each image location, in 25 contrast response values.

We applied each of the 5 scale-specific filters, combined with each of the 5 contrast gain controls, to 800 natural images. Figure 1 shows average responses over all image locations. Contrast is high at small scale and low semi-saturation. It decreases exponentially with scale owing to the peak sensitivity of the filters, which is inversely related to spatial scale. That contrast also decreases with semi-saturation is explained by the fact that the amount of contrast suppression is proportional to the semi-saturation value. From these summary statistics it follows that, although natural image contrast varies considerable within and across scale and contrast gain, the fast majority of natural image contrasts falls above a lower threshold. It is reasonable to assume that the LGN considers contrast below this statistical threshold as noise and only processes contrasts above it, i.e. only processes reliable contrast outputs.

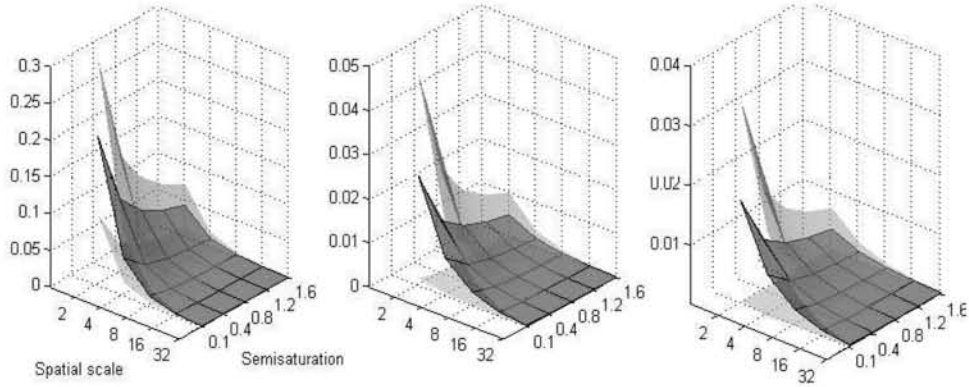

Figure 1: Approximation of the typical range of contrasts generated by LGN neurons tuned to different spatial frequencies (5 octave scales) and with different contrast gain properties (5 semi-saturation constants). Shown are the average of local contrast (dark gray), plus and minus two standard deviations, in the gray level (left), blue-yellow (middle) and red-green (right) color components of 800 natural images.

## 2.2 Natural image statistics-based selection of unique local image contrast values

What spatial scale and contrast gain does the LGN use to process local image contrast? It is unlikely that the LGN (linearly) integrates the output of a population of spatially overlapping filters to determine local image contrast as this would make it sensitive to receptive field clutter [11]. Here we depart from the view that LGN aims to minimize receptive clutter by selecting a single output from a population of scale and gain specific contrast filters [12]. Specifically, in order to determine contrast at an image location, we apply the smallest filter with boosted contrast output above what can be expected to be noise for that particular filter. We define local contrast as the amount of contrast exceeding the noise threshold, which for a given scale and gain is set here to half standard deviation of contrasts in 800 natural images (see figure 1). This contrast response selection mechanism produces a contrast magnitude map in ways similar to the scale selection model in [13].

We apply the local contrast selection mechanism separately to the individual color components of an image. From a single color image, the three color components are extracted using the Gaussian color model [14], resulting in a gray-scale, blue-yellow and red-green image representations. Each of these representations is convolved with the 25 scale and gain specific contrast filters and subsequently subjected to our local contrast selection mechanism. For each color component a dedicated scale and gain dependent noise threshold is used (see figure 1). As a result, for each color image we get three contrast magnitude maps, which we linearly sum to arrive at a single contrast magnitude map.

## 2.3 Weibull statistics of local image contrast

The contrast magnitude map of an image is summarized in a histogram, representing the distribution of local contrast values of that image. Note that the histogram does not preserve information about spatial structure in the contrast magnitude map: a scrambling the contrast magnitude map will not affect the histogram. We subsequently fit a three-parameter Weibull distribution to the contrast histogram. The three-parameter Weibull distribution is given by

$$f(x) = c \exp^{\left(\frac{x-\mu}{\beta}\right)^{\gamma}} \qquad (1)$$

The parameters of this distribution are indicative for the spatial structure in a natural scene (see figure 2) and can be put in a biologically plausible framework [7]. The scale parameter $\beta$ describes the width of the histogram. Hence, it varies roughly with the variation in local image contrasts. The shape parameter $\gamma$ describes the shape of the histogram. It varies with the amount of scene clutter. The $\mu$ parameter, represents the origin of the distribution. Its position is influenced by uneven illumination. The three Weibull parameters are estimated using a maximum likelihood estimator (MLE). To achieve illumination invariance, the $\mu$ parameter is normalized out.

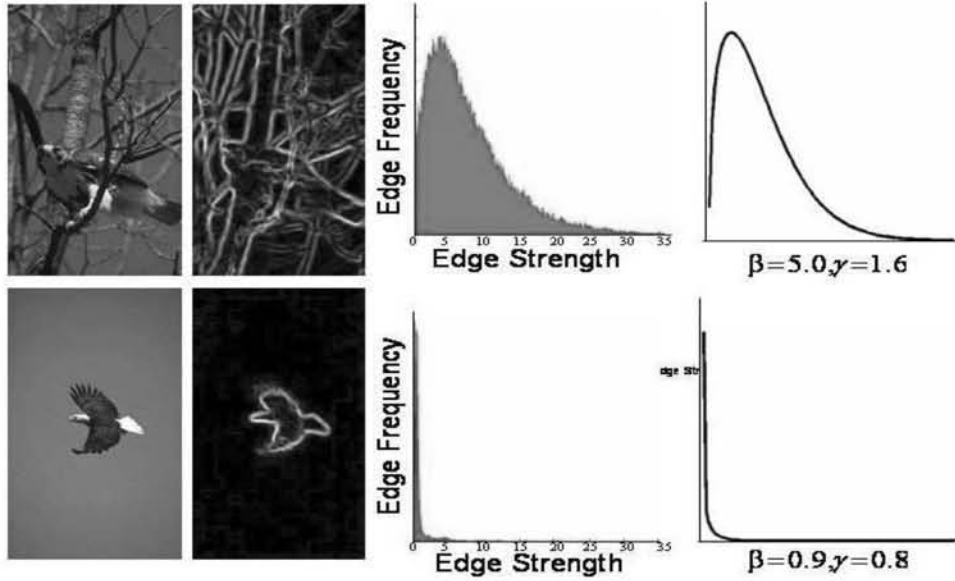

Figure 2: Two arbitrary natural images from the Corel Photo Library with varying degrees of details and varying degrees of scene clutter. The details in the upper image are chaotic. They range from large for the bird to small for partially occluded tree branches. In contrast, the second picture depicts a single coherent object, the eagle, against a highly uniform background. The image gradient at each image location shows the contrast strength. All gradients accumulated in a histogram reveal the distribution of local contrasts. The scale and shape parameters of the Weibull distribution are estimated from the fit to the histogram by maximum likelihood estimation.

## 2.4 Model Estimation

We use EEG response signals from $C$ channels (electrodes) covering the entire cortex, to develop a Weibull response model that predicts neuronal responses to natural images. EEG signals are measured for $S$ subjects watching $N$ natural images. We average these signals across subjects to obtain a more robust response signal per channel and per image. This results in an $N \times C$ matrix $\mathbf{F}(t)$ of response signals $f_{nc}(t)$. We construct a linear Weibull response model for each channel separately. Our rationale for combining the two Weibull parameters in a linear fashion is that these two parameters can be suitably extracted from the X and Y units in the LGN model (as shown in Scholte et al [7]) and as such the linear combination reflects linear pooling at the LGN level.

Functional data analysis [15] provides a natural framework for modeling continuous stochastic brain processes. We use a point-wise multivariate functional linear model to establish the relation between Weibull parameters $\mathbf{X} = [\beta_1, ..., \beta_N; \gamma_1, ..., \gamma_N]^T$ and the EEG response $\mathbf{f}_c(t) = [\bar{f}_{nc}, ..., \bar{f}_{Nc}]^T$. The values $\beta_n, \gamma_n$ are the Weibull parameters of image $n$ and $\bar{f}_{nc}$ is the across subject average response to that image at channel $c$. Weibull response model estimation for channel $c$ then reduces to solving

$$\mathbf{f}_c(t) = \mathbf{X}\omega(t) + \boldsymbol{\epsilon}(t) \tag{2}$$

where $\omega(t)$ is $2 \times 1$ vector of regression functions and $\boldsymbol{\epsilon}(t) = [\epsilon_1(t), ...., \epsilon_S(t)]^T$ is the vector of residual functions. Under the assumption that the residual functions $\boldsymbol{\epsilon}(t)$ are independent and normally distributed with zero mean, the regression function is estimated by least squares minimization such that

$$\hat{\omega}_c(t) = \min_{\omega^*(t)} \int_t \|\mathbf{f}_c(t) - \mathbf{X}\omega^*(t)\|^2 dt. \tag{3}$$

A roughness penalty, based on the second derivative of $\omega(t)$, regularize the estimate $\hat{\omega}_c(t)$. The estimated regression function provides the best estimate of $\mathbf{f}_c(t)$ in least squares sense:

$$\hat{\mathbf{f}}_c(t) = \mathbf{X}\hat{\omega}_c(t). \tag{4}$$

We use $\hat{\omega}_c(t)$ to predict the EEG responses to a new set of $M$ images represented by their Weibull distribution. The EEG responses to these new images are predicted using the Weibull response

model:

$$\hat{\mathbf{g}}_c(t) = \mathbf{Y}\hat{\omega}_c(t). \tag{5}$$

where the $M \times 2$ data matrix $\mathbf{Y}$ contains the two Weibull parameters for each of the new images and the M-vector of functions $\hat{\mathbf{g}}_c(t)$ denotes the predicted neural responses for channel $c$.

## 2.5  Image Identification

How well does the Weibull response model predict EEG responses to natural images? We answer this question in terms of EEG-based identification of individual images. Given a set of $M$ new images and their Weibull parameters $\mathbf{Y}$, the Weibull response model provides the EEG prediction $\hat{\mathbf{g}}_c(t)$. The match between prediction $\hat{\mathbf{g}}_c(t)$ and true, measured EEG activity $\mathbf{g}_c(t) = [g_1, ..., g_M]$ provides a means for image identification. More specifically, an $M \times M$ similarity matrix $\mathbf{S}$ is constructed, where each element contains the Pearson's correlation coefficient $R$ between measured $g_{cm}(t)$ and predicted $\hat{g}_{cm}(t)$ response. The similarity matrix shows for each individual image, the amount of EEG correlation with the other images. The image whose predicted activity pattern is most correlated with the measured activity pattern is selected. A similarity matrix is constructed separately for each of the $C$ channels. These similarity matrices are squared in order to allow averaging of similarity matrices across channels. Hence, the square of the correlation coefficient $r^2$ rather than $r$ itself is used as a measure of similarity between true and predicted response.

# 3  Experiments and Results

## 3.1  Stimulus and EEG Data

In our experiments we used 800 color images with a resolution 345 x 217 pixels and a bit-depth of 24. Of these, 400 were pictures of animals in their natural habitat and 400 pictures of natural landscapes, city scenes, indoor scenes and man-made objects. These images were taken from a larger set of images used in Fabre-Thorpe [16]. This subset of images was reasonably balanced in terms of Michelson contrast, spatial frequency and orientation properties. The Weibull properties of these images nevertheless covered a wide range of real-world images. The data set did not contain near duplicates.

The images were presented to 32 subjects on a 19" Ilyama monitor with a resolution of 1024*768 pixels and a frame-rate of 100 Hz. Subjects were seated 90 cm from the monitor. During EEG acquisition a stimulus was presented, on average every 1500 ms (range 1000-2000 ms) for 100 ms. Each stimulus was presented 2 times for a total of 1600 presentations. Recordings were made with a Biosemi 52-channel Active Two EEG system (Biosemi Instrumentation BV, Amsterdam, The Netherlands). Data was sampled at 256 Hz. Data analysis was identical to [17] with the exception that the high-pass filter was placed at 0.1 Hz (12 db/octave) and the pre-stimulus baseline activity was taken between -100 and 0 ms with regard to stimulus onset. Trials were averaged over subject per individual stimulus resulting in 800 averages of 20 to 32 averages per individual image.

## 3.2  Experiments

The experiments were carried out with the following parameters settings. Two banks of Gaussian second-order derivative filters were used to determine image contrast for each image location. The first set consisted of filters with octave spatial scales 1.5, 3, 6, 12, 24 (std. in pixels). This set was used to determine the Weibull scale parameter $\beta$. The other filter bank, with scales 3, 6, 12, 24, 48, was used for the estimation of Weibull shape parameter $\gamma$. The spatial properties of the two sets were determined experimentally and roughly correspond to receptive field sizes of small X and large Y Ganglion cells in the early visual system of the human brain [18]. We used 5 semi-saturation constants between 0.15 and 1.6 to cover the spectrum from linear to non-linear contrast gain control in the LGN.

A cross validation study was performed to obtain reliable performance measurements. We repeated the same experiment 50 times, each time randomly selecting 700 images for model estimation and 100 images for image identification. Performance was measured in terms of the percentage of correctly identified images for each of the 50 experiments. The 50 measures were then averaged to

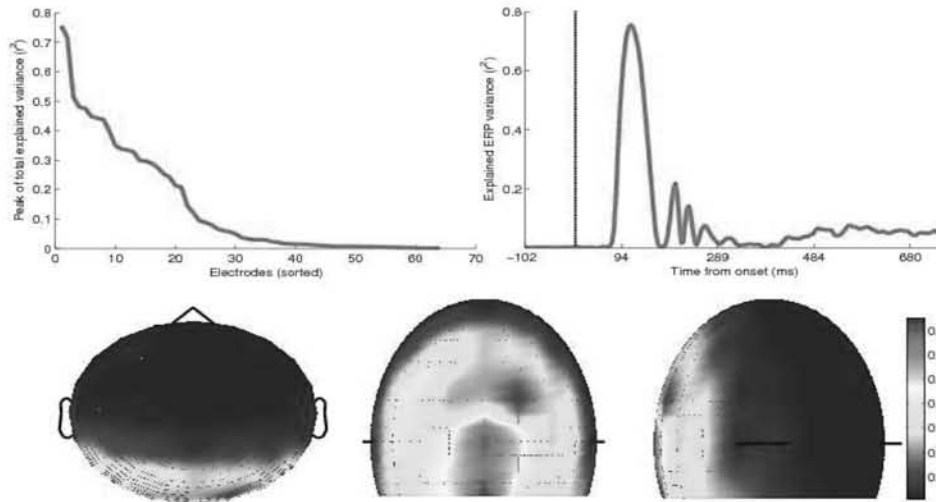

Figure 3: Total explained variance in ERP signals by the two Weibull parameters. The peak of the total explained variance is highest (75 percent) for the IZ electrode overlying the early visual cortex and gradually decays at higher brain areas. The time course of explained variance for the IZ electrode reveals that the peak occurs at 113 ms after stimulus onset.

arrive at a single performance outcome. Hence, accuracy was defined as the fraction of images for which the predicted activity pattern and measured activity pattern produced the highest $r^2$. As accuracy does not reflect how close the correct image was to being selected, we also ranked the correlation coefficients and determined within which percentage of the ranked $M$ images the correct one was.

## 3.3 Results

We first present correlations between ERP signals from across the entire brain and the two parameters of the Weibull fit to the sum of selected local contrast values in the gray-level, blue-yellow and red-green components of each image. Correlations are strikingly high at electrode Iz overlying the early visual cortex. The peak $r^2$ (square of the correlation coefficient) over time for that electrode is 75 percent (r = 0.8691; p = 0). The peak $r^2$ over time slowly decays away from the occipital part of the head as can be seen from the topographic plots in figure 3. The Weibull parameters explain most variance in the ERP signal very early in visual processing at 113 ms after stimulus onset (3) and continue to explain variance up to about 200 ms. This suggests that the two Weibull parameters are probably only relevant to the brain in the early phases of visual processing.

Accuracy results are shown in figure 4. The topographic plots show image identification accuracy for single channels (electrodes). Channel IZ produces the highest accuracy with 5 percent. This means that based on ERP signal at the IZ electrode, 5 out of 100 images are on average correctly identified from the similarity matrix. Then follow channel Oz with 4.3 percent, O2 with 4.1 and so on. Image identification based on multiple channels strikingly improves performance as shown in figure 4. When the similarity matrices from the 20 most contributive channels are averaged, accuracy of almost 90 percent is obtained. This means that, with a Weibull response model of only two parameters, almost every image can be correctly identified from the neural activity that this image triggers. As an aside we note that this implies that the different parts of the early visual system process different types of images (in terms of the two Weibull parameters) in different ways.

To test the individual contribution of the Weibull parameters, we performed principal component analysis on the beta and gamma parameters and used the principal component scores separately for image identification. A Weibull response model based only on one of the two principal component scores performs significantly less as can be seen in figure 4. Moreover, there is large difference in accuracy performance between the two projected Weibull parameters. These results demonstrate

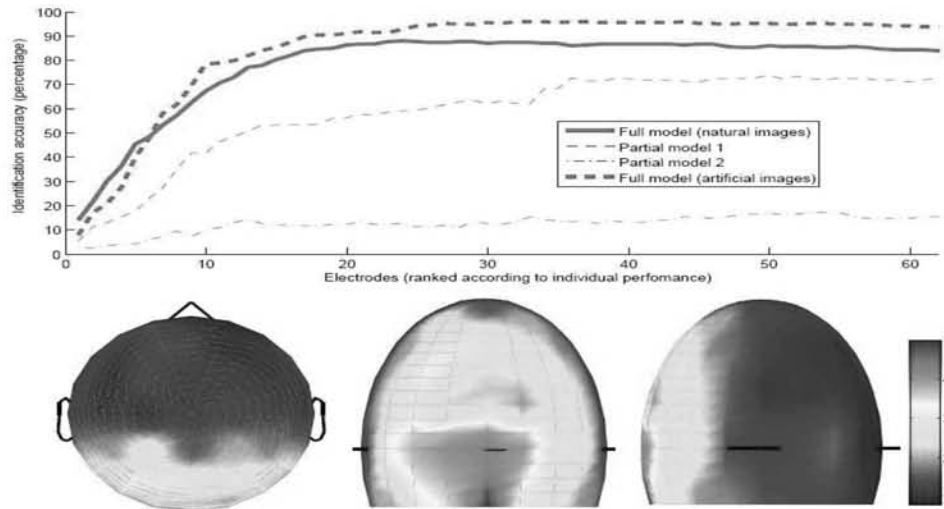

Figure 4: Accuracy performance for the full (two-parameter) and partial (orthogonal projection of one of the two parameters) Weibull response model. Accuracy is based on the accumulation of image identification at multiple channels. The topographic plots show the accuracy performance for the individual channels.

that the two Weibull parameters indeed capture two different perceptual aspects of natural scenes, which together constitutes an important part of early neural processing of natural images.

Accuracy results in figure 4 only show how often the correct image is ranked first, not where it is ranked. We therefore analyzed the image rankings (data not shown). For the first most contributive channel (41), the correct image is always ranked within the top 13 percent of the images. The ranking slightly worsens (top 15 percent) for the second most contributive channel (Oz) and for the third (O2, top 16 percent). From the fourth channel and beyond there is a clear but steady drop in ranking. The ranking data show an overall pattern similar to the one seen in the accuracy data and indicate that, even in cases where an image is not correctly identified, the misidentification is limited.

When does identification fail? We extracted frequently confused image pairs from all similarity matrices of all 50 cross validation steps for all 64 channels. These image pairs reveal that identification errors tend to occur when the selected image is visually similar to the correct image. The upper row of figure 5 shows 4 images from our data set and the images with which these have been confused frequently. The first set of 2 images, containing grazing cows and a packed donkey, have been confused 6 times across the 50 cross validation experiments, the second, and third set 5 times and the fourth set 4 times. The overall similarity between the confused images is evident and remarkable considering the variety of images we have used. These findings suggest that the Weibull model captures aspect of a scene's visual gist that the brain possibly uses for perception at a glance.

We further scrutinized image identification performance on occlusion models of natural images. Following [19], we created 24 types of dead leave images containing disks of various sizes (large, medium and small), size distributions (Power law and exponential), intensities (equal intensity versus decaying intensity) and opacities (occluding versus transparent). For each image type, 16 instances were composed resulting in a total of 384 dead leave images. We presented 16 subjects with the 384 dead leaves images while recording their EEG activity. As with our natural images, the beta and gamma parameter values of the Weibull contrast distributions underlying the 364 dead leave images correlated highly with EEG activity ($r^2 = 0.83$). A cross validation experiment in which we used 284 dead leaves images for building a Weibull response model and 100 for image identification resulted in an average image identification performance of 94 percent (see figure 4). Confusion analysis revealed that dead leave images with clear disks were well identified, whereas dead leaves images composed of transparent and thus indistinguishable disks were confused frequently (figure

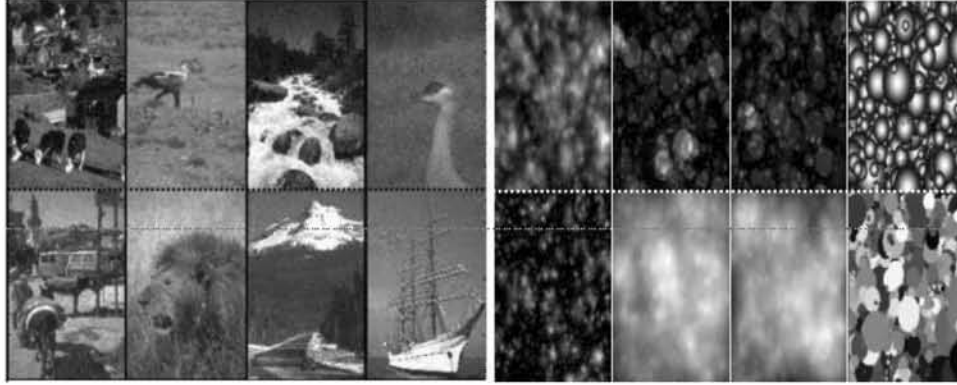

Figure 5: Most confused image pairs during cross-validation. Note the global similarity in spatial configuration between the natural image pairs. Similarity between most confused dead leave image pairs is also apparent: except for the fourth pair, they are all images with transparent disk (but with different disk sizes and disk intensity patterns). Dead leave images with small, opaque and equal intensity disks (as in the lower right example) were least confused.

5). Apparently, the information in the EEG signal that facilitates image identification is related to clear object-background differences.

## 4 Discussion and Conclusion

To determine local image contrasts, we have applied a bank of biologically-motivated contrast filters to each image location and selected a single filter output based on receptive field size and response reliability. The statistics of locally selected image contrasts, appropriately captured by the Weibull distribution, explain up to 75 percent of occipital EEG activity for natural images and almost 83 for artificial dead leave images. We have used Weibull contrast statistics of these images and corresponding EEG activity to construct a Weibull response model for EEG-based rapid image identification. Using this model, we have obtained image identification performance of 90 percent for natural images and 94 percent for dead leave images, which is remarkable considering the simplicity of the two-parameter Weibull image model and the limited spatial resolution of EEG data. We attribute this success to the ability of the Weibull parameters to structure the space of natural images in a highly meaningful and compact way, invariant to a large class of accidental or trivial scene features. Both the scale and shape parameters contribute to the meaningful organization of natural images and appear to play an important role in the early neural processing of natural images.

Kay et. al [8] report similar image identification performance using an other biologically plausible model. In this model, a natural image is represented by a large set of Gabor wavelets differing in size, position, orientation, spatial frequency and phase. Haemodynaymic responses in the visual cortex are integrally modeled as a linear function of the contrast energy contained in quadrature wavelet pairs. In a repeated trial experiment involving 1700 training images, 120 test images, and fMRI data of 2 subjects, 92 percent of the test images were correctly identified for one subject and 72 for a second subject. In a single trial experiment, the reported performances are 52 and 31 percent respectively. We note that in contrast to [8], our neural response model is based on (summary) statistics of filter outputs, rather than on filter outputs themselves. This may explain our models ability to compactly describe a scene's visual gist.

In conclusion, we embrace the view that common factors of natural images imprinted in the brain daily, underlie rapid image identification by humans. Departing from this view, we establish a relationship between natural image statistics and neural processing through the Weibull response model. Results with EEG-based image identification using the Weibull response model, together with the biological plausibility of the Weibull response model, supports the idea that the human visual system evolved, among others, to estimate the Weibull statistics of natural images for rapid extraction of their visual gist [7].

# References

[1] E.P. Simoncelli and Olshausen. B. Natural image statistics and neural representation. *Annu. Rev. Neurosci.*, 24:11931216, 2001.

[2] A. Oliva and A. Torralba. Building the gist of a scene: The role of global image features in recognition. *Visual Perception, Progress in Brain Research*, 155, 2006.

[3] S. G. Mallat. A theory for multiresolution signal decomposition: The wavelet representation. *IEEE Trans. Pattern Anal. Mach. Intell.*, 11(7):674–693, 1989.

[4] M.A. Thomson. Higher-order structure in natural scenes. *J. Opt. Soc. Am. A*, 16:15491553, 1999.

[5] E.P. Simoncelli, A. Srivastava, A.B. Lee, and S-C Zhu. On advances in statistical modeling of natural images. *Journal of Mathematical Imaging and Vision*, 18(1), 2003.

[6] J. M. Geusebroek and A. W. M. Smeulders. A six-stimulus theory for stochastic texture. *International Journal of Computer Vision*, 62(1/2):7–16, 2005.

[7] H. S. Scholte, S. Ghebreab, A. Smeulders, and V. Lamme. Brain responses strongly correlate with weibull image statistics when processing natural images. *Journal of Vision*, 9(4):1–15, 2009.

[8] K.N. Kay, T. Naselaris, R.J. Prenger, and J.L. Gallant. Identifying natural images from human brain activity. *Nature*, 452:352–355, 2008.

[9] L.J. Croner and E. Kaplan. Receptive fields of p and m ganglion cells across the primate retina. *Vision Research*, 35(11):7–24, 1995.

[10] V. Bonin, V. Mante, and M. Carandini. The suppressive field of neurons in lateral geniculate nucleus. *Journal of Neuroscience*, 25:10844–10856, 2005.

[11] M. Riesenhuber and T. Poggio. Hierarchical models of object recognition in cortex. *Nature Neuroscience*, 2(11):1019–1025, November 1999.

[12] S.Ghebreab, H.S Scholte, V.A.F. Lamme, and A.W.M. Smeulders. Neural adaption to the spatial distribution of multi-scale contrast in natural images. *Submitted*.

[13] J.H. Elder and S.W. Zucker. Local scale control for edge detection and blur estimation. *IEEE Transactions on Pattern Analysis and Machine intelligence*, 20:699–716, 1998.

[14] J. M. Geusebroek, R. van den Boomgaard, A. W. M. Smeulders, and H. Geerts. Color invariance. *IEEE Transactions on Pattern Analysis and Machine intelligence*, 23(12):1338–1350, 2001.

[15] J. Ramsay and B. Silverman. *Functional Data Analysis*. Springer-Verlag, 1997.

[16] M. Fabre-Thorpe, A. Delorme, C. Marlot, and S.J. Thorpe. A limit to the speed of processing in ultra-rapid visual categorisation of novel natural scenes. *Journal Cognitive Neuroscience*, 13:171–180, 2001.

[17] J.J. Fahrenfort, H.S. Scholte, and V.A.F. Lamme. The spatiotemporal profile of cortical processing leading up to visual perception. *Journal of Vision*, 8(1):1–12, 2008.

[18] L. Watanabe and R.W. Rodieck. Parasol and midget ganglion cells of the primate retina. *Journal of Computational Neurology*, 289:434–454, 1989.

[19] W.H. Hsiao and R.P. Millane. Effects of occlusion, edges, and scaling on the power spectra of natural images. *J. Opt. Soc. Am. A*, 22:1789–1797, 2005.
